# Synergies between Intrinsic and Synaptic Plasticity in Individual Model Neurons

**Jochen Triesch**
Dept. of Cognitive Science, UC San Diego, La Jolla, CA, 92093-0515, USA
Frankfurt Institute for Advanced Studies, Frankfurt am Main, Germany
`triesch@ucsd.edu`

## Abstract

This paper explores the computational consequences of simultaneous intrinsic and synaptic plasticity in individual model neurons. It proposes a new intrinsic plasticity mechanism for a continuous activation model neuron based on low order moments of the neuron's firing rate distribution. The goal of the intrinsic plasticity mechanism is to enforce a sparse distribution of the neuron's activity level. In conjunction with Hebbian learning at the neuron's synapses, the neuron is shown to discover sparse directions in the input.

## 1 Introduction

Neurons in the primate visual system exhibit a sparse distribution of firing rates. In particular, neurons in different visual cortical areas show an approximately exponential distribution of their firing rates in response to stimulation with natural video sequences [1]. The brain may do this because the exponential distribution maximizes entropy under the constraint of a fixed mean firing rate. The fixed mean firing rate constraint is often considered to reflect a desired level of metabolic costs. This view is theoretically appealing. However, it is currently not clear how neurons adjust their firing rate distribution to become sparse. Several different mechanisms seem to play a role: First, synaptic learning can change a neuron's response to a distribution of inputs. Second, intrinsic learning may change conductances in the dendrites and soma to adapt the distribution of firing rates [7]. Third, non-linear lateral interactions in a network can make a neuron's responses more sparse [8]. In the extreme case this leads to winner-take-all networks, which form a code where only a single unit is active for any given stimulus. Such ultra-sparse codes are considered inefficient, however. This paper investigates the interaction of intrinsic and synaptic learning processes in individual model neurons in the learning of sparse codes.

We consider an individual continuous activation model neuron with a non-linear transfer function that has adjustable parameters. We are proposing a simple intrinsic learning mechanism based on estimates of low-order moments of the activity distribution that allows the model neuron to adjust the parameters of its non-linear transfer function to obtain an approximately exponential distribution of its activity. We then show that if combined with a standard Hebbian learning rule employing multiplicative weight normalization, this leads to the extraction of sparse features from the input. This is in sharp contrast to standard Hebbian learning in linear units with multiplicative weight normalization, which leads to

the extraction of the principal Eigenvector of the input correlation matrix. We demonstrate the behavior of the combined intrinsic and synaptic learning mechanisms on the classic bars problem [4], a non-linear independent component analysis problem.

The remainder of this paper is organized as follows. Section 2 introduces our scheme for intrinsic plasticity and presents experiments demonstrating the effectiveness of the proposed mechanism for inducing a sparse firing rate distribution. Section 3 studies the combination of intrinsic plasticity with Hebbian learning at the synapses and demonstrates how it gives rise to the discovery of sparse directions in the input. Finally, Sect. 4 discusses our findings in the context of related work.

## 2   Intrinsic Plasticity Mechanism

Biological neurons do not only adapt synaptic properties but also change their excitability through the modification of voltage gated channels. Such *intrinsic* plasticity has been observed across many species and brain areas [9]. Although our understanding of these processes and their underlying mechanisms remains quite unclear, it has been hypothesized that this form of plasticity contributes to a neuron's *homeostasis* of its mean firing rate level. Our basic hypothesis is that the goal of intrinsic plasticity is to ensure an approximately exponential distribution of firing rate levels in individual neurons. To our knowledge, this idea was first investigated in [7], where a Hodgkin-Huxley style model with a number of voltage gated conductances was considered. A learning rule was derived that adapts the properties of voltage gated channels to match the firing rate distribution of the unit to a desired distribution. In order to facilitate the simulation of potentially large networks we choose a different, more abstract level of modeling employing a continuous activation unit with a non-linear transfer function. Our model neuron is described by:

$$Y = S_\theta(X) , \quad X = \mathbf{w}^T \mathbf{u} , \tag{1}$$

where $Y$ is the neuron's output (firing rate), $X$ is the neuron's total synaptic current, $\mathbf{w}$ is the neuron's weight vector representing synaptic strengths, the vector $\mathbf{u}$ represents the pre-synaptic input, and $S_\theta(.)$ is the neuron's non-linear transfer function (activation function), parameterized by a vector of parameters $\theta$. In this section we will not be concerned with synaptic mechanism changing the weight vector $\mathbf{w}$, so we will just consider a particular distribution $p(X = x) \equiv p(x)$ of the net synaptic current and consider the resulting distribution of firing rates $p(Y = y) \equiv p(y)$. Intrinsic plasticity is modeled as inducing changes to the non-linear transfer function with the goal of bringing the distribution of activity levels $p(y)$ close to an exponential distribution.

In general terms, the problem is that of matching a distribution to another. Given a signal with a certain distribution, find a non-linear transfer function that converts the signal to one with a desired distribution. In image processing, this is typically called histogram matching. If there are no restrictions on the non-linearity then a solution can always be found. The standard example is histogram equalization, where a signal is passed through its own cumulative density function to give a uniform distribution over the interval $[0, 1]$. While this approach offers a general solution, it is unclear how individual neurons could achieve this goal. In particular, it requires that the individual neuron can change its non-linear transfer function arbitrarily, i.e. it requires infinitely many degrees of freedom.

### 2.1   Intrinsic Plasticity Based on Low Order Moments of Firing Rate

In contrast to the general scheme outlined above the approach proposed here utilizes a simple sigmoid non-linearity with only two adjustable parameters $a$ and $b$:

$$S_{ab}(X) = \frac{1}{1 + \exp\left(-\left(X - b\right)/a\right)} . \tag{2}$$

Parameter $a > 0$ changes the steepness of the sigmoid, while parameter $b$ shifts it left/right[1]. Qualitatively similar changes in spike threshold and slope of the activation function have been observed in cortical neurons. Since the non-linearity has only two degrees of freedom it is generally not possible to ascertain an exponential activity distribution for an arbitrary input distribution. A plausible alternative goal is to just match low order moments of the activity distribution to those of a specific target distribution. Since our sigmoid non-linearity has two parameters, we consider the first and second moments.

For a random variable $T$ following an exponential distribution with mean $\mu$ we have:

$$p(T = t) = \frac{1}{\mu} \exp\left(-t/\mu\right) \; ; \quad M_T^1 \equiv \langle T \rangle = \mu \; ; \quad M_T^2 \equiv \langle T^2 \rangle = 2\mu^2 \; , \qquad (3)$$

where $\langle . \rangle$ denotes the expected value operator. Our intrinsic plasticity rule is formulated as a set of simple proportional control laws for $a$ and $b$ that drive the first and second moments $\langle Y \rangle$ and $\langle Y^2 \rangle$ of the output distributions to the values of the corresponding moments of an exponential distribution $M_T^1$ and $M_T^2$:

$$\dot{a} = \gamma \left(\langle Y^2 \rangle - 2\mu^2\right) \; , \quad \dot{b} = \eta \left(\langle Y \rangle - \mu\right) \; , \qquad (4)$$

where $\gamma$ and $\eta$ are learning rates. The mean $\mu$ of the desired exponential distribution is a free parameter which may vary across cortical areas. Equations (4) describe a system of coupled integro-differential equations where the integration is implicit in the expected value operations. Note that both $\langle Y \rangle$ and $\langle Y^2 \rangle$ depend on the sigmoid parameters $a$ and $b$. From (4) it is obvious that there is a stationary point of these dynamics if the first and second moment of $Y$ equal the desired values of $\mu$ and $2\mu^2$, respectively.

The first and second moments of $Y$ need to be estimated online. In our model, we calculate estimates $\hat{M}_Y^1$ and $\hat{M}_Y^2$ of $\langle Y \rangle$ and $\langle Y^2 \rangle$ according to:

$$\dot{\hat{M}}_Y^1 = \lambda(y - \hat{M}_Y^1) \; , \quad \dot{\hat{M}}_Y^2 = \lambda(y^2 - \hat{M}_Y^2) \; , \qquad (5)$$

where $\lambda$ is a small learning rate.

## 2.2 Experiments with Intrinsic Plasticity Mechanism

We tested the proposed intrinsic plasticity mechanism for a number of distributions of the synaptic current $X$ (Fig. 1). Consider the case where this current follows a Gaussian distribution with zero mean and unit variance: $X \sim \mathcal{N}(0, 1)$. Under this assumption we can calculate the moments $\langle Y \rangle$ and $\langle Y^2 \rangle$ (although only numerically) for any particular values of $a$ and $b$. Panel a in Fig. 1 shows a phase diagram of this system. Its flow field is sketched and two sample trajectories converging to a stationary point are given. The stationary point is at the intersection of the nullclines where $\langle Y \rangle = \mu$ and $\langle Y^2 \rangle = 2\mu^2$. Its coordinates are $a_\infty \approx 0.90$, $b_\infty \approx 2.38$. Panel b compares the theoretically optimal transfer function (dotted), which would lead to an exactly exponential distribution of $Y$, with the learned sigmoidal transfer function (solid). The learned transfer function gives a very good fit. The resulting distribution of $Y$ is in fact very close to the desired exponential distribution. For the general case of a Gaussian input distribution with mean $\mu_G$ and standard deviation $\sigma_G$, the sigmoid parameters will converge to $a \to a_\infty \sigma_G$ and $b \to b_\infty \sigma_G + \mu_G$ under the intrinsic plasticity rule. If the input to the unit can be assumed to be Gaussian, this relation can be used to calculate the desired parameters of the sigmoid non-linearity directly.

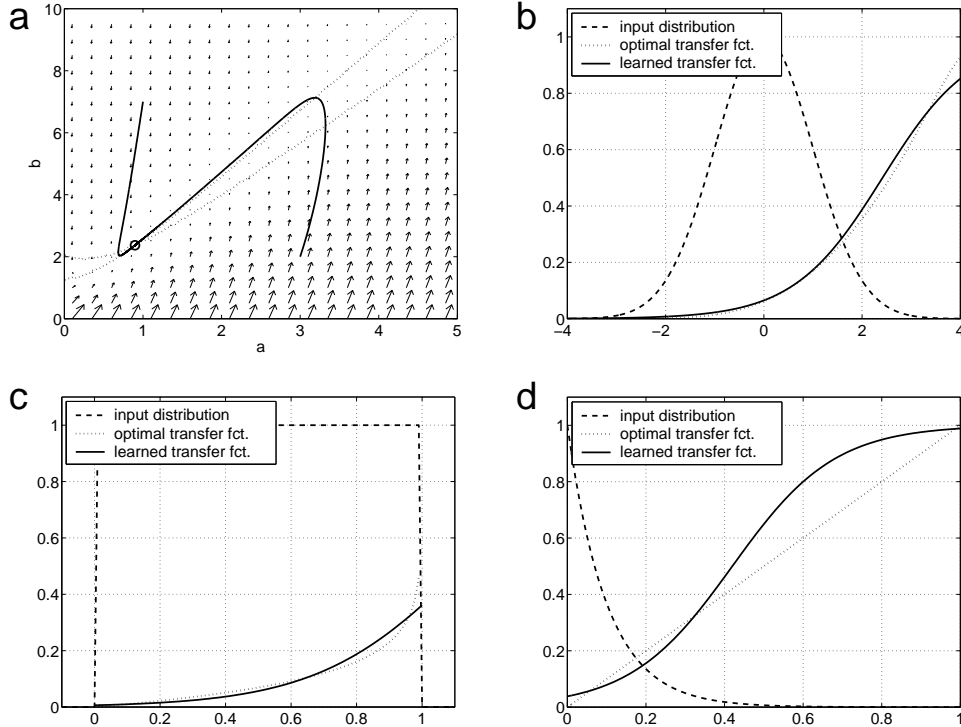

Figure 1: Dynamics of intrinsic plasticity mechanism for various input distributions. **a,b:** Gaussian input distribution. Panel **a** shows the phase plane diagram. Arrows indicate the flow field of the system. Dotted lines indicate approximate locations of the nullclines (found numerically).Two example trajectories are exhibited which converge to the stationary point (marked with a circle). Panel **b** shows the optimal (dotted) and learned transfer function (solid). The Gaussian input distribution (dashed, not drawn to scale) is also shown. **c,d:** same as **b** but for uniform and exponential input distribution. Parameters were $\mu = 0.1, \lambda = 5 \times 10^{-4}, \eta = 2 \times 10^{-3}, \gamma = 10^{-3}$.

Panels c and d show the result of intrinsic plasticity for two other input distributions. In the case of a uniform input distribution in the interval $[0, 1]$ (panel c) the optimal transfer function becomes infinitely steep for $x \rightarrow 1$. For an exponentially distributed input (panel d), the ideal transfer function would simply be the identity function. In both cases the intrinsic plasticity mechanism adjusts the sigmoid non-linearity in a sensible fashion and the output distribution is a fair approximation of the desired exponential distribution.

## 2.3 Discussion of the Intrinsic Plasticity Mechanism

The proposed mechanism for intrinsic plasticity is effective in driving a neuron to exhibit an approximately exponential distribution of firing rates as observed in biological neurons in the visual system. The general idea is not restricted to the use of a sigmoid non-linearity. The same adaptation mechanism can also be used in conjunction with, say, an adjustable threshold-linear activation function. An interesting alternative to the proposed mechanism can be derived by directly minimizing the KL divergence between the output distribution and the desired exponential distribution through stochastic gradient descent. The resulting learning rule, which is closely related to a rule for adapting a sigmoid nonlinearity to max-

imize the output entropy derived by Bell and Sejnowski[2], will be discussed elsewhere. It leads to very similar results to the ones presented here.

A biological implementation of the proposed mechanism is plausible. All that is needed are estimates of the first and second moment of the firing rate distribution. A specific, testable prediction of the simple model is that changes to the distribution of a neuron's firing rate levels that keep the average firing rate of the neuron unchanged but alter the second moment of the firing rate distribution should lead to measurable changes in the neuron's excitability.

# 3    Combination of Intrinsic and Synaptic Plasticity

In this Section we want to study the effects of simultaneous intrinsic and synaptic learning for an individual model neuron. Synaptic learning is typically modeled with Hebbian learning rules, of which a large number are being used in the literature. In principle, any Hebbian learning rule can be combined with our scheme for intrinsic plasticity. Due to space limitations, we only consider the simplest of all Hebbian learning rules:

$$\Delta \mathbf{w} = \alpha \mathbf{u} Y(\mathbf{u}) = \alpha \mathbf{u} S_{ab}(\mathbf{w}^T \mathbf{u}) \,, \tag{6}$$

where the notation is identical to that of Sec. 2 and $\alpha$ is a learning rate. This learning rule is unstable and needs to be accompanied by a scheme limiting weight growth. We simply adopt a multiplicative normalization scheme that after each update re-scales the weight vector to unit length: $\mathbf{w} \leftarrow \mathbf{w}/\| \mathbf{w} \|$.

## 3.1    Analysis for the Limiting Case of Fast Intrinsic Plasticity

Under a few assumptions, an interesting intuition about the simultaneous intrinsic and Hebbian learning can be gained. Consider the limit of intrinsic plasticity being much faster than Hebbian plasticity. This may not be very plausible biologically, but it allows for an interesting analysis. In this case we may assume that the non-linearity has adapted to give an approximately exponential distribution of the firing rate $Y$ before $\mathbf{w}$ can change much. Thus, from (6), $\Delta \mathbf{w}$ can be seen as a weighted sum of the inputs $\mathbf{u}$, with the activities $Y$ acting as weights that follow an approximately exponential distribution. Since similar inputs $\mathbf{u}$ will produce similar outputs $Y$, the expected value of the weight update $\langle \Delta \mathbf{w} \rangle$ will be dominated by a small set of inputs that produce the highest output activities. The remainder of the inputs will "pull" the weight vector back to the average input $\langle \mathbf{u} \rangle$. Due to the multiplicative weight normalization, the stationary states of the weight vector are reached if $\Delta \mathbf{w}$ is parallel to $\mathbf{w}$, i.e., if $\langle \Delta \mathbf{w} \rangle = k \mathbf{w}$ for some constant $k$.

A simple example shall illustrate the effect of intrinsic plasticity on Hebbian learning in more detail. Consider the case where there are only two clusters of inputs at the locations $\mathbf{c}_1$ and $\mathbf{c}_2$. Let us also assume that both clusters account for exactly half of the inputs. If the weight vector is slightly closer to one of the two clusters, inputs from this cluster will activate the unit more strongly and will exert a stronger "pull" on the weight vector. Let $m = \mu \ln(2)$ denote the median of the exponential firing rate distribution with mean $\mu$. Then inputs from the closer cluster, say $\mathbf{c}_1$, will be responsible for all activities above $m$ while the inputs from the other cluster will be responsible for all activities below $m$. Hence, the expected value of the weight update $\langle \Delta \mathbf{w} \rangle$ will be given by:

$$\langle \Delta \mathbf{w} \rangle \approx \alpha \mathbf{c}_1 \int_m^\infty \frac{y}{\mu} \exp(-y/\mu) dy + \alpha \mathbf{c}_2 \int_0^m \frac{y}{\mu} \exp(-y/\mu) dy \tag{7}$$

$$= \frac{\alpha \mu}{2} \left( (1 + \ln 2)\, \mathbf{c}_1 + (1 - \ln 2)\, \mathbf{c}_2 \right) \,. \tag{8}$$

Taking the multiplicative weight normalization into account, we see that the weight vector

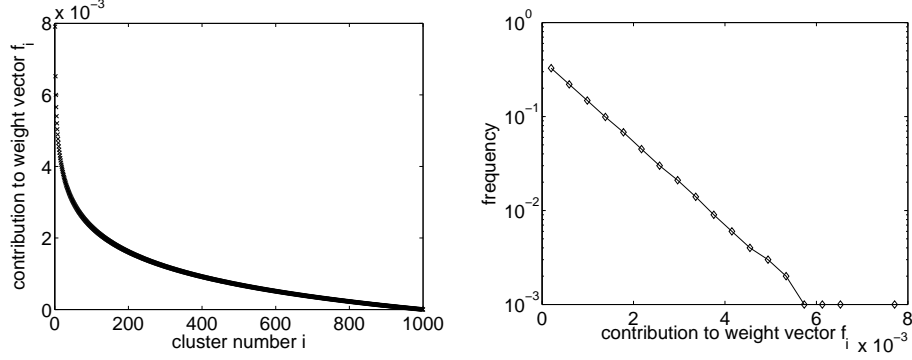

Figure 2: Left: relative contributions to the weight vector $f_i$ for $N = 1000$ input clusters (sorted). Right: the distribution of the $f_i$ is approximately exponential.

will converge to either of the following two stationary states:

$$\mathbf{w} = \frac{(1 \pm \ln 2)\mathbf{c_1} + (1 \mp \ln 2)\mathbf{c_2}}{\| (1 \pm \ln 2)\mathbf{c_1} + (1 \mp \ln 2)\mathbf{c_2} \|} \ . \tag{9}$$

The weight vector moves close to one of the two clusters but does not fully commit to it.

For the general case of $N$ input clusters, only a few clusters will strongly contribute to the final weight vector. Generalizing the result from above, it is not difficult to derive that the weight vector $\mathbf{w}$ will be proportional to a weighted sum of the cluster centers:

$$\mathbf{w} \propto \sum_{i=1}^{N} f_i \mathbf{c_i} \ ; \ \text{with } f_i = 1 + \log(N) - i \log(i) + (i - 1) \log(i - 1) \ , \tag{10}$$

where we define $0 \log(0) \equiv 0$. Here, $f_i$ denotes the relative contribution of the $i$-th closest input cluster to the final weight vector. There can be at most $N!$ resulting weight vectors owing to the number of possible assignments of the $f_i$ to the clusters. Note that the final weight vector does not depend on the desired mean activity level $\mu$. Fig. 2 plots (10) for $N = 1000$ (left) and shows that the resulting distribution of the $f_i$ is approximately exponential (right).

We can see why such a weight vector may correspond to a sparse direction in the input space as follows: consider the case where the input cluster centers are random vectors of unit length in a high-dimensional space. It is a property of high-dimensional spaces that random vectors are approximately orthogonal, so that $\mathbf{c}_i^T \mathbf{c}_j \approx \delta_{ij}$, where $\delta_{ij}$ is the Kronecker delta. If we consider the projection of an input from an arbitrary cluster, say $\mathbf{c}_j$, onto the weight vector, we see that $\mathbf{w}^T \mathbf{c}_j \propto \left( \sum_i f_i \mathbf{c}_i^T \right) \mathbf{c}_j \approx f_j$. The distribution of $X = \mathbf{w}^T \mathbf{u}$ follows the distribution of the $f_i$, which is approximately exponential. Thus, the projection of all inputs onto the weight vector has an approximately exponential distribution. Note that this behavior is markedly different from Hebbian learning in a linear unit which leads to the extraction of the principal eigenvector of the input correlation matrix.

It is interesting to note that in this situation the optimal transfer function $S^*$ that will make the unit's activity $Y$ have an exponential distribution of a desired mean $\mu$ is simply a multiplication with a constant $k$, i.e. $S^*(X) = kX$. Thus, depending on the initial weight vector and the resulting distribution of $X$, the neuron's activation function may transiently adapt to enforce an approximately exponential firing rate distribution, but the simultaneous Hebbian learning drives it back to a linear form. In the end, a simple linear activation function may result from this interplay of intrinsic and synaptic plasticity. In fact, the observation of approximately linear activation functions in cortical neurons is not uncommon.

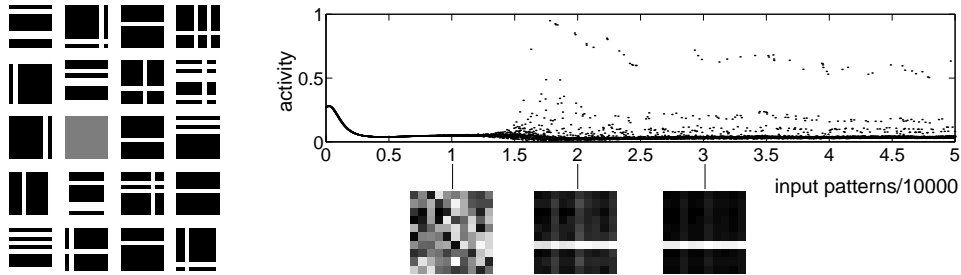

Figure 3: Left: example stimuli from the "bars" problem for a 10 by 10 pixel retina. Right: the activity record shows the unit's response to every 10th input pattern. Below, we show the learned weight vector after presentation of 10,000, 20,000, and 30,000 training patterns.

### 3.2 Application to the "Bars" Problem

The "bars" problem is a standard problem for unsupervised learning architectures [4]. It is a non-linear ICA problem for which traditional ICA approaches have been shown to fail [5]. The input domain consists of an $N$-by-$N$ retina. On this retina, all horizontal and vertical bars ($2N$ in total) can be displayed. The presence or absence of each bar is determined independently, with every bar occurring with the same probability $p$ (in our case $p = 1/N$). If a horizontal and a vertical bar overlap, the pixel at the intersection point will be just as bright as any other pixels on the bars, rather than twice as bright. This makes the problem a non-linear ICA problem. Example stimuli from the bars dataset are shown in Fig. 3 (left). Note that we normalize input vectors to unit length. The goal of learning in the bars problem is to find the independent sources of the images, i.e., the individual bars. Thus, the neural learning system should develop filters that represent the individual bars.

We have trained an individual sigmoidal model neuron on the bars input domain. The theoretical analysis above assumed that intrinsic plasticity is much faster than synaptic plasticity. Here, we set the intrinsic plasticity to be slower than the synaptic plasticity, which is more plausible biologically, to see if this may still allow the discovery of sparse directions in the input. As illustrated in Fig. 3 (right) the unit's weight vector aligns with one of the individual bars as soon as the intrinsic plasticity has pushed the model neuron into a regime where its responses are sparse: the unit has discovered one of the independent sources of the input domain. This result is robust if the desired mean activity $\mu$ of the unit is changed over a wide range. If $\mu$ is reduced from its default value ($1/2N = 0.05$) over several orders of magnitude (we tried down to $10^{-5}$) the result remains unchanged. However, if $\mu$ is increased above about $0.15$, the unit will fail to represent an individual bar but will learn a mixture of two or more bars, with different bars being represented with different strengths. Thus, in this example — in contrast to the theoretical result above — the desired mean activity $\mu$ does influence the weight vector that is being learned. The reason for this is that the intrinsic plasticity only imperfectly adjusts the output distribution to the desired exponential shape. As can be seen in Fig. 3 the output has a multimodal structure. For low $\mu$, only the highest mode, which corresponds to a specific single bar presented in isolation, contributes strongly to the weight vector.

## 4 Discussion

Biological neurons are highly adaptive computation devices. While the plasticity of a neuron's synapses has always been a core topic of neural computation research, there has been little work investigating the computational properties of intrinsic plasticity mechanisms and

the relation between intrinsic and synaptic learning. This paper has investigated the potential role of intrinsic learning mechanisms operating at the soma when used in conjunction with Hebbian learning at the synapses. To this end, we have proposed a new intrinsic plasticity mechanism that adjusts the parameters of a sigmoid nonlinearity to move the neuron's firing rate distribution to a sparse regime. The learning mechanism is effective in producing approximately exponential firing rate distributions as observed in neurons in the visual system of cats and primates. Studying simultaneous intrinsic and synaptic learning, we found a synergistic relation between the two. We demonstrated how the two mechanisms may cooperate to discover sparse directions in the input. When applied to the classic "bars" problem, a single unit was shown to discover one of the independent sources as soon as the intrinsic plasticity moved the unit's activity distribution into a sparse regime. Thus, this research is related to other work in the area of Hebbian projection pursuit and Hebbian ICA, e.g., [3, 6]. In such approaches, the "standard" Hebbian weight update rule is modified to allow the discovery of non-gaussian directions in the input. We have shown that the combination of intrinsic plasticity with the standard Hebbian learning rule can be sufficient for the discovery of sparse directions in the input. Future work will analyze the combination of intrinsic plasticity with other Hebbian learning rules. Further, we would like to consider networks of such units and the formation of map-like representations. The nonlinear nature of the transfer function may facilitate the construction of hierarchical networks for unsupervised learning. It will also be interesting to study the effects of intrinsic plasticity in the context of recurrent networks, where it may contribute to keeping the network in a certain desired dynamic regime.

### Acknowledgments

The author is supported by the National Science Foundation under grants NSF 0208451 and NSF 0233200. I thank Erik Murphy-Chutorian and Emanuel Todorov for discussions and comments on earlier drafts.

## Footnotes

[1]Note that while we view adjusting $a$ and $b$ as changing the shape of the sigmoid non-linearity, an equivalent view is that $a$ and $b$ are used to linearly rescale the signal $X$ before it is passed through a "standard" logistic function. In general, however, intrinsic plasticity may give rise to non-linear changes that cannot be captured by such a linear re-scaling of all weights.

## References

[1] R. Baddeley, L. F. Abbott, M.C. Booth, F. Sengpiel, and T. Freeman. Responses of neurons in primary and inferior temporal visual cortices to natural scenes. *Proc. R. Soc. London, Ser. B*, 264:1775–1783, 1998.

[2] A. J. Bell and T. J. Sejnowski. An information-maximization approach to blind separation and blind deconvolution. *Neural Computation*, 7:1129–1159, 1995.

[3] B. S. Blais, N. Intrator, H. Shouval, and L. N. Cooper. Receptive field formation in natural scene environments. *Neural Computation*, 10:1797–1813, 1998.

[4] P. Földiák. Forming sparse representations by local anti-hebbian learning. *Biological Cybernetics*, 64:165–170, 1990.

[5] S. Hochreiter and J. Schmidhuber. Feature extraction through LOCOCODE. *Neural Computation*, 11(3):679–714, 1999.

[6] A. Hyvärinen and E. Oja. Independent component analysis by general nonlinear hebbian-like learning rules. *Signal Processing*, 64(3):301–313, 1998.

[7] M. Stemmler and C. Koch. How voltage-dependent conductances can adapt to maximize the information encoded by neuronal firing rate. *Nature Neuroscience*, 2(6):521–527, 1999.

[8] W. E. Vinje and J. L. Gallant. Sparse coding and decorrelation in primary visual cortex during natural vision. *Science*, 287:1273–1276, 2000.

[9] W. Zhang and D. J. Linden. The other side of the engram: Experience-driven changes in neuronal intrinsic excitability. *Nature Reviews Neuroscience*, 4:885–900, 2003.
